# A NETWORK FOR IMAGE SEGMENTATION USING COLOR

Anya Hurlbert and Tomaso Poggio
Center for Biological Information Processing at Whitaker College
Department of Brain and Cognitive Science
and the MIT AI Laboratory
Cambridge, MA 02139
(hurlbert@wheaties.ai.mit.edu)

## ABSTRACT

We propose a parallel network of simple processors to find color boundaries irrespective of spatial changes in illumination, and to spread uniform colors within marked regions.

## INTRODUCTION

To rely on color as a cue in recognizing objects, a visual system must have at least approximate color constancy. Otherwise it might ascribe different characteristics to the same object under different lights. But the first step in using color for recognition, segmenting the scene into regions of different colors, does not require color constancy. In this crucial step color serves simply as a means of distinguishing one object from another in a given scene. Color differences, which mark material boundaries, are essential, while absolute color values are not. The goal of segmentation algorithms is to achieve this first step toward object recognition by finding discontinuities in the image irradiance that mark material boundaries.

The problems that segmentation algorithms must solve is how to choose color labels, how to distinguish material boundaries from other changes in the image that give rise to color edges, and how to fill in uniform regions with the appropriate color labels. (Ideally, the color labels should remain constant under changes in the illumination or scene composition and color edges should occur only at material boundaries.) Rubin and Richards (1984) show that algorithms can solve the second problem under some conditions by comparing the image irradiance signal in distinct spectral channels on either side of an edge.

The goal of the segmentation algorithms we discuss here is to find boundaries between regions of different surface spectral reflectances and to spread uniform colors within them, without explicitly requiring the colors to be constant under changes in illumination. The color labels we use are analogous to the CIE chromaticity coordinates $x$ and $y$. Under the single source assumption, they change across space

only when the surface spectral reflectance changes, except when strong specularities are present. (The algorithms therefore require help at a later stage to identify between color label changes due to specularities, which we have not yet explicitly incorporated.) The color edges themselves are localised with the help of luminance edges, by analogy with psychophysics of segmentation and filling-in. The Koffka Ring illusion, for example, indicates that color is attributed to surfaces by an interaction between an edge-finding operator and a filling-in operator.[1] The interaction is justified by the fact that in the real world changes in surface spectral reflectance are almost always accompanied by changes in brightness.

## Color Labels

We assume that surfaces reflect light according to the *neutral-interface-reflection* model. In this model (Lee, 1986 , Shaefer, 1984 [3]) the image irradiance $I(x, y, \lambda)$ is the sum of two components, the *surface* reflection and the *body* reflection:

$$I(x, y, \lambda) = L(\mathbf{r}(x, y), \lambda)[a(\mathbf{r}, \lambda)g(\delta(\mathbf{r})) + bh(\delta(\mathbf{r}))],$$

where $\lambda$ labels wavelength and $\mathbf{r}(x, y)$ is the point on the 3D surface to which the image coordinates $(x, y)$ correspond. $L(\mathbf{r}(x, y), \lambda)$ is the illumination on the surface. $a(\mathbf{r}, \lambda)$ is the spectral reflectance factor of the body reflection component and $g(\delta(\mathbf{r}))$ its magnitude, which depends on the viewing geometry parameters lumped together in $\delta(\mathbf{r})$. The spectral reflectance factor of the specular, or surface reflection, component $b$ is assumed to be constant with respect to $\lambda$, as is true for inhomogeneous materials such as paints and plastics. For most materials, the magnitude of the specular component $h$ depends strongly on the viewing geometry. Using the single source assumption, we may factor the illumination $L$ into separate spatial and spectral components $(L(\mathbf{r}, \lambda) = L(\mathbf{r})c(\lambda))$. Multiplying $I$ by the spectral sensitivities of the color sensors $i = 1, 2, 3$ and integrating over wavelength yields the triplet of color values $(R, G, B)$, where

$$R = I^R(x, y) = L(\mathbf{r}(x, y))(a^R(\mathbf{r}(x, y))g(\delta) + b^R h(\delta))$$

and so forth and where the $a^i$ and $b^i$ are the reflectance factors in the spectral channels defined by the sensor spectral sensitivities.

We define the hues $u$ and $v$ as

$$u = \frac{R}{R + G + B}$$

and

$$v = \frac{G}{R+G+B}$$

at each pixel.

In Lambertian reflection, the specular reflectance factor $b$ is zero. In this case, $u$ and $v$ are piecewise constant: they change in the image only when the $a^i(x,y)$ change. Thus $u$ or $v$ mark discontinuities in the surface spectral reflectance function, e.g they mark material boundaries. Conversely, image regions of constant $u$ correspond to regions of constant surface color. Synthetic images generated with standard computer graphics algorithms (using, for example, the Phong reflectance model) behave in this way: $u$ is constant across the visible surface of a shaded sphere. Across specularities, $u$ in general changes but often not much. Thus one approach to the segmentation problem is to find regions of "constant" $u$ and their boundaries.

The difficulty with this approach is that real $u$ data are noisy and unreliable: $u$ is the quotient of numbers that are not only noisy themselves but also, at least for biological photosensor spectral sensitivities, very close to one another. The goals of segmentation algorithms are therefore to enhance discontinuities in $u$ and, within the regions marked by the discontinuities, to smoothe over the noise and fill in the data where they are unreliable. We have explored several methods of meeting these goals.

## Segmentation Algorithms

One method is to regularize – to eliminate the noise and fill in the data, while preserving the discontinuities. Using an algorithm based on Markov Random Field techniques, we have obtained encouraging results on real images (see Poggio et al., 1988). The MRF technique exploits the constraint that $u$ should be *piecewise constant* within the discontinuity contours and uses image brightness edges as guides in finding the contours.

An alternative to the MRF approach is a cooperative network that fills in data and filters out noise while enforcing the constraint of piecewise constancy. The network, a type of Hopfield net, is similar to the cooperative stereo network of Marr and Poggio (1976). Another approach consists of a one-pass winner-take-all scheme. Both algorithms involve loading the initial hue values into discrete bins, an undesirable and biologically unlikely feature. Although they produce good results on noisy synthetic images and can be improved by modification (see Hurlbert, 1989), another class of algorithms which we now describe are simple and effective, especially on parallel computers such as the Connection Machine.

## Averaging Network

One way to avoid small step changes in hue across a uniform surface resulting from initial loading into discrete bins is to relax the local requirement for piecewise

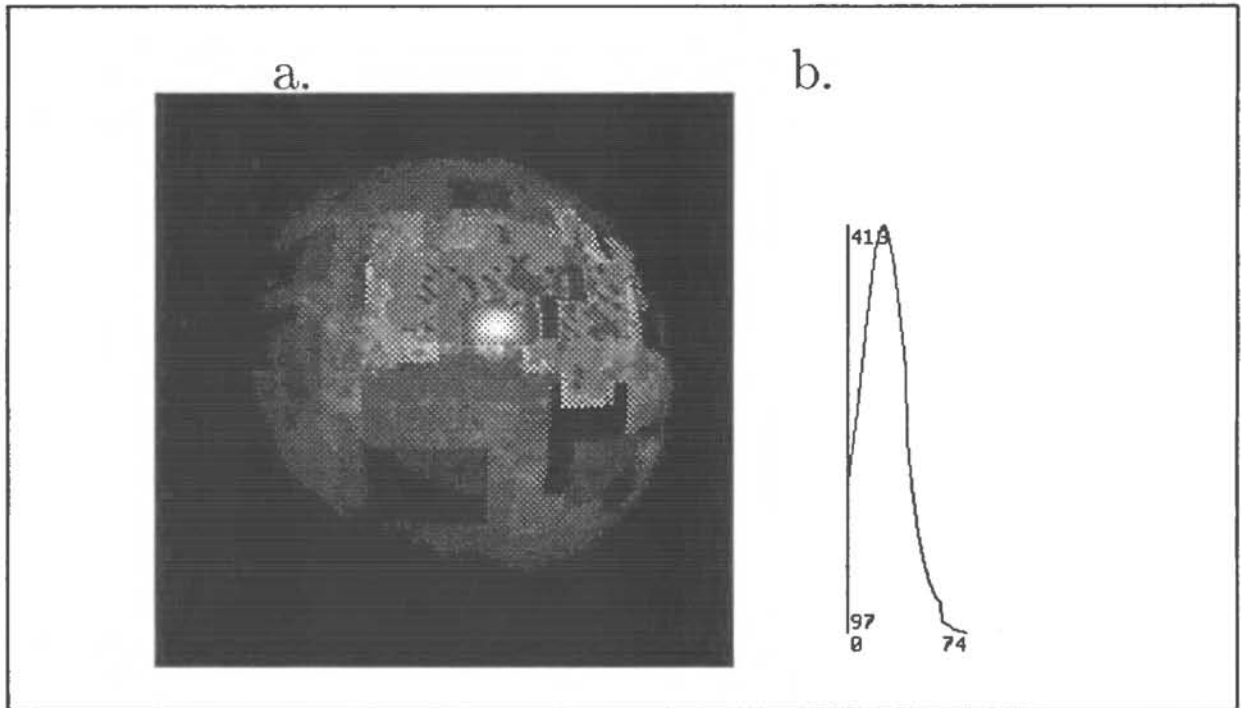

Figure 1: (a) Image of a Mondrian-textured sphere – the red channel. (b) Vertical slice through the specularity in a 75 x 75 pixel region of the three-channel image $(R + G + B)$ of the same sphere.

constancy and instead require only that hue be smooth within regions delineated by the edge input. We will see that this local smoothness requirement actually yields an iterative algorithm that provides asymptotically *piecewise constant* hue regions.

To implement the local smoothness criterion we use an averaging scheme that simply replaces the value of each pixel in the hue image with the average of its local surround, iterating many times over the whole image.

The algorithm takes as input the hue image (either the $u$-image or the $v$-image) and one or two edge images, either luminance edges alone, or luminance edges plus $u$ or $v$ edges, or $u$ edges plus $v$ edges. The edge images are obtained by performing Canny edge detection or by using a thresholded directional first derivative. On each iteration, the value at each pixel in the hue image is replaced by the average of its value and those in its contributing neighborhood. A neighboring pixel is allowed to contribute if (i) it is one of the four pixels sharing a full border with the central pixel (ii) it shares the same edge label with the central pixel in all input edge images (iii) its value is non-zero and (iv) its value is within a fixed range of the central pixel value. The last requirement simply reinforces the edge label requirement when a hue image serves as an input edge image – the edge label requirement allows only those pixels that lie on the same side of an edge to be averaged, while the other insures that only those pixels with similar hues are averaged.

More formally

$$h_{i,j}^{n+1} = \frac{1}{N(C^n)} \sum_{l,m \in C(h_{i,j}^n)} h_{l,m}^n$$

where $C^n(h_{i,j}^n)$ is the set of $N(C^n)$ pixels among the next neighbors of $i,j$ that differ from $h_{i,j}^n$ less than a specified amount and are not crossed by an edge in the edge map(s) (on the assumption that the pixel $(i,j)$ does not belong to an edge). The iteration of this operator is similar to nonlinear diffusion and to discontinuous regularization of the type discussed by Blake and Zisserman (1987), Geman and Geman (1984) and Marroquin [9]. The iterative scheme of the above equation can be derived from minimization via gradient descent of the energy function

$$E = \sum E_{i,j}$$

with

$$E_{i,j} = (1 - d_{i+1,j})V(h_{i,j}, h_{i+1,j}) + (1 - d_{i,j+1})V(h_{i,j}, h_{i,j+1})$$

$$+(1 - d_{i-1,j})V(h_{i,j}, h_{i-1,j}) + (1 - d_{i,j-1})V(h_{i,j}, h_{i,j-1}),$$

where $V(x,y) = V(x-y)$ is a quadratic potential around 0 and constant for $|x-y|$ above a certain value.

The local averaging smoothes noise in the hue values and spreads uniform hues across regions marked by the edge inputs. On images with shading but without strong specularities the algorithm performs a clean segmentation into regions of different hues.

## Conclusions

The averaging scheme finds constant hue regions under the assumptions of a single source and no strong specularities. A strong highlight may originate an edge that could then "break" the averaging operation. In our limited experience most specularities seem to average out and disappear from the smoothed hue map, largely because even strong specularities in the image are much reduced in the initial hue image. The iterative averaging scheme completely eliminates the remaining gradients in hue. It is possible that more powerful discrimination of specularities will require specialized routines and higher-level knowledge (Hurlbert, 1989).

Yet this simple network alone is sufficient to reproduce some psychophysical phenomena. In particular, the interaction between brightness and color edges enables the network to mimic such visual "illusions" as the Koffka Ring. We replicate the illusion in the following way. A black-and-white Koffka Ring (a uniform grey annulus against a rectangular bipartite background, one side black and the other white) (Hurlbert and Poggio, 1988b) is filtered through the lightness filter estimated in

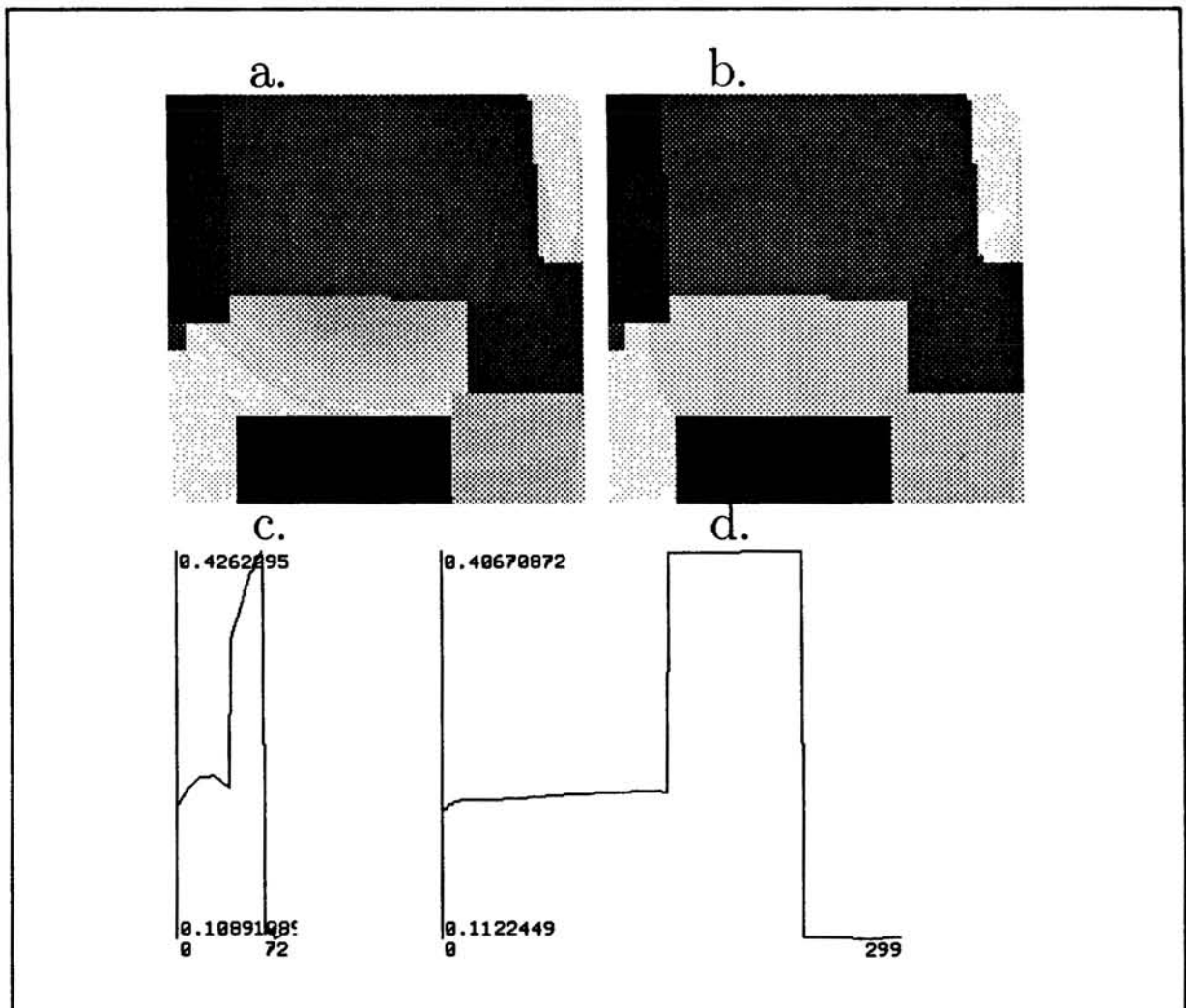

Figure 2: (a) A 75x75 pixel region of the u image, including the specularity. (b) The image obtained after 500 iterations of the averaging network on (a), using as edge input the Canny edges of the luminance image. A threshold on differences in the v image allows only similar v values to be averaged. (c) Vertical slice through center of (a). (d) Vertical slice at same coordinates through (b) (note different scales of (c) and (d)).

the way described elsewhere (Hurlbert and Poggio, 1988a). (For black-and-white images this step replaces the operation of obtaining $u$ and $v$: in both cases the goal is to eliminate spatial gradients of in the effective illumination.) The filtered Koffka Ring is then fed to the averaging network together with the brightness edges. When in the input image the boundary between the two parts of the background continues across the annulus, in the output image (after 2000 iterations of the averaging network) the annulus splits into two semi-annuli of different colors in the output image, dark grey against the white half, light grey against the black half (Hurlbert, 1989). When the boundary does not continue across the annulus, the annulus remains a uniform grey. These results agree with human perception.

# Acknowledgements

This report describes research done within the Center for Biological Information Processing, in the Department of Brain and Cognitive Sciences, and at the Artificial Intelligence Laboratory. This research is sponsored by a grant from the Office of Naval Research (ONR), Cognitive and Neural Sciences Division; by the Artificial Intelligence Center of Hughes Aircraft Corporation; by the Alfred P. Sloan Foundation; by the National Science Foundation; by the Artificial Intelligence Center of Hughes Aircraft Corporation (S1-801534-2); and by the NATO Scientific Affairs Division (0403/87). Support for the A. I. Laboratory's artificial intelligence research is provided by the Advanced Research Projects Agency of the Department of Defense under Army contract DACA76–85–C–0010, and in part by ONR contract N00014–85–K–0124. Tomaso Poggio is supported by the Uncas and Helen Whitaker Chair at the Massachusetts Institute of Technology, Whitaker College.

**References**

John Rubin and Whitman Richards. Colour vision: representing material categories. Artificial Intelligence Laboratory Memo 764, Massachusetts Institute of Technology, 1984.

Hsien-Che Lee. Method for computing the scene-illuminant chromaticity from specular highlights. *Journal of the Optical Society of America*, 3:1694–1699, 1986.

Steven A. Shafer. Using color to separate reflection components. *Color Research and Applications*, 10(4):210–218, 1985.

Tomaso Poggio, J. Little, E. Gamble, W. Gillett, D. Geiger, D. Weinshall, M. Villalba, N. Larson, T. Cass, H. Bülthoff, M. Drumheller, P. Oppenheimer, W. Yang, and A. Hurlbert. The MIT Vision Machine. In *Proceedings Image Understanding Workshop*, Cambridge, MA, April 1988. Morgan Kaufmann, San Mateo, CA.

David Marr and Tomaso Poggio. Cooperative computation of stereo disparity. *Science*, 194:283–287, 1976.

Anya C. Hurlbert. *The Computation of Color*. PhD thesis, Massachusetts Institute of Technology, Cambridge, MA, 1989.

Jose L. Marroquin. *Probabilistic Solution of Inverse Problems*. PhD thesis, Massachusetts Institute of Technology, Cambridge, MA, 1985.

Andrew Blake and Andrew Zisserman. *Visual Reconstruction.* MIT Press, Cambridge, Mass, 1987.

Stuart Geman and Don Geman. Stochastic relaxation, Gibbs distributions, and the Bayesian restoration of images. *IEEE Transactions on Pattern Analysis and Machine Intelligence,* PAMI-6:721–741, 1984.

Anya C. Hurlbert and Tomaso A. Poggio. Learning a color algorithm from examples. In Dana Z. Anderson, editor, *Neural Information Processing Systems.* American Institute of Physics, 1988.

A. C. Hurlbert and T. A. Poggio. Synthesizing a color algorithm from examples. *Science,* 239:482–485, 1988.

## Footnotes

[1] Note that Land's original retinex algorithm, which thresholds and sums the differences in image irradiance between adjacent points along many paths, accounts for the contribution of edges to color, without introducing a separate luminance edge detector.
